# Derivative observations in Gaussian Process Models of Dynamic Systems

**E. Solak**
Dept. Elec. & Electr. Eng.,
Strathclyde University,
Glasgow G1 1QE,
Scotland, UK.
*ercan.solak@strath.ac.uk*

**R. Murray-Smith**[1,2]
[1]Dept. Computing Science,
University of Glasgow
Glasgow G12 8QQ,
Scotland, UK.
*rod@dcs.gla.ac.uk*

**W. E. Leithead**
[2]Hamilton Institute,
National Univ. of
Ireland, Maynooth,
Co. Kildare, Ireland.
*bill@icu.strath.ac.uk*

**D. J. Leith**
Hamilton Institute,
National Univ. of
Ireland, Maynooth,
Co. Kildare, Ireland
*doug.leith@may.ie*

**C. E. Rasmussen**
Gatsby Computational Neuroscience Unit,
University College London, UK
*edward@gatsby.ucl.ac.uk*

## Abstract

Gaussian processes provide an approach to nonparametric modelling which allows a straightforward combination of function and derivative observations in an empirical model. This is of particular importance in identification of nonlinear dynamic systems from experimental data. 1) It allows us to combine derivative information, and associated uncertainty with normal function observations into the learning and inference process. This derivative information can be in the form of priors specified by an expert or identified from perturbation data close to equilibrium. 2) It allows a seamless fusion of multiple local linear models in a consistent manner, inferring consistent models and ensuring that integrability constraints are met. 3) It improves dramatically the computational efficiency of Gaussian process models for dynamic system identification, by summarising large quantities of near-equilibrium data by a handful of linearisations, reducing the training set size – traditionally a problem for Gaussian process models.

## 1 Introduction

In many applications which involve modelling an unknown system $y = f(\mathbf{x})$ from observed data, model accuracy could be improved by using not only observations of $y$, but also observations of derivatives e.g. $\partial y / \partial x_i$. These derivative observations might be directly available from sensors which, for example, measure velocity or acceleration rather than position, they might be prior linearisation models from historical experiments. A further practical reason is related to the fact that the computational expense of Gaussian processes increases rapidly ($O(N^3)$) with training set size $N$. We may therefore wish to

use linearisations, which are cheap to estimate, to describe the system in those areas in which they are sufficiently accurate, efficiently summarising a large subset of training data. We focus on application of such models in modelling nonlinear dynamic systems from experimental data.

## 2 Gaussian processes and derivative processes

### 2.1 Gaussian processes

Bayesian regression based on Gaussian processes is described by [1] and interest has grown since publication of [2, 3, 4]. Assume a set $S$ of input/output pairs, $\{(\mathbf{x}^i, y^i)\}$ are given, where $\mathbf{x}^i \in \mathbf{R}^D$, $y^i \in \mathbf{R}$, $i = 1 \ldots N$. In the GP framework, the output values $y_i$ are viewed as being drawn from a zero-mean multivariable Gaussian distribution whose covariance matrix is a function of the input vectors $\mathbf{x}^i$. Namely the output distribution is

$$(y^1, \ldots, y^N | \mathbf{x}^1, \ldots, \mathbf{x}^N) \sim \mathcal{N}(0, \Lambda(S, S)).$$

A general model, which reflects the higher correlation between spatially close (in some appropriate metric) points – a smoothness assumption in target system $f(x)$ – uses a covariance matrix with the following structure;

$$\Lambda_{ij}(S, S) = \alpha \exp(-\frac{1}{2}\|\mathbf{x}^i - \mathbf{x}^j\|_\Gamma^2) + v\delta_{i,j}, \tag{1}$$

where the norm $\| \cdot \|_\Gamma$ is defined as

$$\|\mathbf{u}\|_\Gamma = (u'\Gamma u)^{\frac{1}{2}}, \;\; \Gamma = \mathrm{diag}(\gamma_1, \ldots, \gamma_D).$$

The $D + 2$ variables, $\alpha, \gamma_1, \ldots, \gamma_D, v$ are the *hyper-parameters* of the GP model, which are constrained to be non-negative. In particular $v$ is included to capture the noise component of the covariance. The GP model can be used to calculate the distribution of an unknown output $y^{N+1}$ corresponding to known input $\mathbf{x}^{N+1}$ as

$$(y^{N+1} | \mathbf{x}^1, \ldots, \mathbf{x}^N, \mathbf{x}^{N+1}, y^1, \ldots, y^N) \sim \mathcal{N}(\mu, \bar{\Lambda}),$$

where

$$\mu = \Lambda(\mathbf{x}^{N+1}, S)\Lambda^{-1}(S, S)\mathbf{y}, \tag{2}$$

$$\bar{\Lambda} = \Lambda(\mathbf{x}^{N+1}, \mathbf{x}^{N+1}) - \Lambda(\mathbf{x}^{N+1}, S)\Lambda^{-1}(S, S)\Lambda(S, \mathbf{x}^{N+1}) \tag{3}$$

and $\mathbf{y} = [y^1, y^2, \ldots, y^N]'$.

The mean $\mu$ of this distribution can be chosen as the maximum-likelihood prediction for the output corresponding to the input $\mathbf{x}^{N+1}$.

### 2.2 Gaussian process derivatives

Differentiation is a linear operation, so the derivative of a Gaussian process remains a Gaussian process. The use of derivative observations in Gaussian processes is described in [5, 6], and in engineering applications in [7, 8, 9]. Suppose we are given new sets of pairs $S_j' = \{(\mathbf{x}^{j,i}, \omega^{j,i})\}$, $j = 1, \ldots, D$, $i = 1, \ldots K$, each $S_j'$ corresponding to the $K$ points of $j^{th}$ partial derivative of the underlying function $y = f(\mathbf{x})$. In the noise-free setting this corresponds to the relation

$$\omega^{j,i} = \frac{\partial f(\mathbf{x})}{\partial x_j}\big|_{\mathbf{x}=\mathbf{x}^{j,i}}, \; i = 1, \ldots, K.$$

We now wish to find the joint probability of the vector of $y$'s and $\omega$'s, which involves calculation of the covariance between the function and the derivative observations as well as the covariance among the derivative observations. Covariance functions are typically differentiable, so the covariance between a derivative and function observation and the one between two derivative points satisfy

$$\text{cov}(\omega^{j,m}, y^n) = \frac{\partial}{\partial x_j} \text{cov}(y^m, y^n) \text{ and } \text{cov}(\omega^{j,m}, \omega^{i,n}) = \frac{\partial^2}{\partial x_j \partial x_i} \text{cov}(y^m, y^n).$$

The following identities give those relations necessary to form the full covariance matrix, for the covariance function (1),

$$\text{cov}(y^m, y^n) = \alpha \exp(-\frac{1}{2}\|\mathbf{x}^m - \mathbf{x}^n\|_\Gamma^2) \tag{4}$$

$$\text{cov}(\omega^{j,m}, y^n) = -\alpha\gamma_j(x_j^m - x_j^n) \exp(-\frac{1}{2}\|\mathbf{x}^m - \mathbf{x}^n\|_\Gamma^2), \tag{5}$$

$$\text{cov}(\omega^{j,m}, \omega^{i,n}) = \alpha\gamma_j(\delta_{j,i} - \gamma_i(x_j^m - x_j^n)(x_i^m - x_i^n)) \exp(-\frac{1}{2}\|\mathbf{x}^m - \mathbf{x}^n\|_\Gamma^2) \tag{6}$$

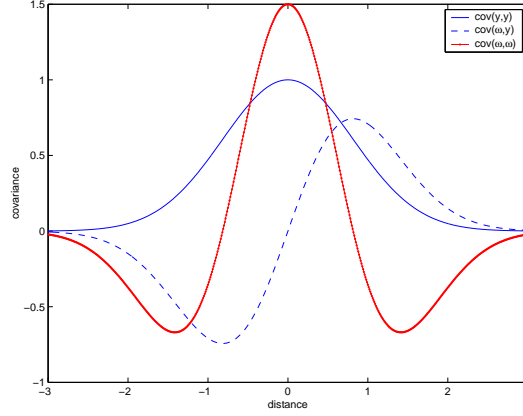

Figure 1: The covariance functions between function and derivative points in one dimension, with hyper-parameters $\gamma_1 = 1.5, \alpha = 1$. The function $\text{cov}(y^m, y^n)$ defines a covariance that decays monotonically as the distance between the corresponding input points $\mathbf{x}^m$ and $\mathbf{x}^n$ increases. Covariance $\text{cov}(\omega^{j,m}, y^n)$ between a derivative point and a function point is an odd function, and does not decrease as fast due to the presence of the multiplicative distance term. $\text{cov}(\omega, \omega)$ illustrates the implicit assumption in the choice of the basic covariance function, that gradients increase with $\gamma$ and that the slopes of realisations will tend to have highest negative correlation at a distance of $x = \sqrt{3/\gamma}$, giving an indication of the typical size of 'wiggles' in realisations of the corresponding Gaussian process .

## 2.3   Derivative observations from identified linearisations

Given perturbation data $x_\delta = x - x_0$, around an equilibrium point $x_{i,0}, y_0$, we can identify a linearisation $\hat{y} = [1 \; x_\delta]\theta_i$, the parameters $\theta_{i,2} \ldots \theta_{i,D+1}$ of which can be viewed as observations of derivatives $\omega^{i,1} \ldots \omega^{i,D}$, and the bias term from the linearisation can be used as a function 'observation', i.e. $\theta_{i,1} = \hat{y}_0$. We use standard linear regression solutions, to estimate the derivatives with a prior of $P$ on the covariance matrix

$$\theta_i = (\mathbf{x_i^T}\mathbf{x_i} + \mathbf{P}^{-1})^{-1}\mathbf{x_i^T}\mathbf{y_i}, \tag{7}$$

$$\sigma_i^2 \quad = \quad \frac{1}{N}(\mathbf{y_i} - \mathbf{x_i}\theta_i^T)^2, \tag{8}$$

$$\Sigma_{\theta_i} \quad = \quad \sigma_i^2 \left(\mathbf{x}_i^T \mathbf{x}_i + P^{-1}\right)^{-1}, \tag{9}$$

$\theta_i$ can be viewed as 'observations' which have uncertainty specified by the a $(D + 1) \times (D + 1)$ covariance matrix $\Sigma_{\theta_i}$ for the $i$th derivative observations, and their associated linearisation point.

With a suitable ordering of the observations (e.g. $\left[y^1 \omega^{1,1} \ldots \omega^{D,1} y^2 \omega^{1,2} \ldots \omega^{D,2}\right]^T$), the associated noise covariance matrix $\Sigma$, which is added to the covariance matrix calculated using (4)-(6), will be block diagonal, where the blocks are the $\Sigma_{\theta_1} \ldots \Sigma_{\theta_K}$ matrices. Use of numerical estimates from linearisations makes it easy to use the full covariance matrix, including off-diagonal elements. This would be much more involved if $\Sigma$ were to be estimated simultaneously with other covariance function hyperparameters.

In a one-dimensional case, given zero noise on observations then two function observations close together give exactly the same information, and constrain the model in the same way as a derivative observation with zero uncertainty. Data is, however, rarely noise-free, and the fact that we can so easily include knowledge of derivative or function observation uncertainty is a major benefit of the Gaussian process prior approach.

The identified derivative and function observation, and their covariance matrix can locally summarise the large number of perturbation training points, leading to a significant reduction in data needed during Gaussian process inference. We can, however, choose to improve robustness by retaining any data in the training set from the equilibrium region which have a low likelihood given the GP model based only on the linearisations (e.g. responses three standard deviations away from the mean).

In this paper we choose the hyper-parameters that maximise the likelihood of the occurrence of the data in the sets $S, S_1', \ldots, S_D'$., using standard optimisation software. Given the data sets $S, S_1', \ldots, S_D'$ and the hyper-parameters the Gaussian process can be used to infer the conditional distribution of the output as well as its partial derivatives for a given input. The ability to predict not only the mean function response, and derivatives but also to be able to predict the input-dependent variance of the function response and derivatives has great utility in the many engineering applications including optimisation and control which depend on derivative information.

### 2.4 Derivative and prediction uncertainty

Figure 2(c) gives intuitive insight into the constraining effect of function observations, and function+derivative observations on realisations drawn from a Gaussian process prior. To further illustrate the effect of knowledge of derivative information on prediction uncertainty. We consider a simple example with a single pair of function observations $(x = 0, y = 1)$ and a single derivative pair $(x = 0, \omega = 0)$. Hyper-parameters are fixed at $\alpha = 1, \gamma_1 = 1, v = 0.01$. Figure 2(a) plots the standard deviation $\sigma$ from models resulting from variations of function and derivatives observations. The four cases considered are

1. a single function observation,

2. a single function observation + a derivative observation, noise-free, i.e. $\Sigma_\omega = 0$,

3. 150 noisy function observations with std. dev. $\sigma_y = 0.02$.

4. a single function observation + uncertain derivative observation (identified from the 150 noisy function observations above, with $\sigma_\omega = 0.2936$, $\theta = [0.9983\ 0.0845]$).

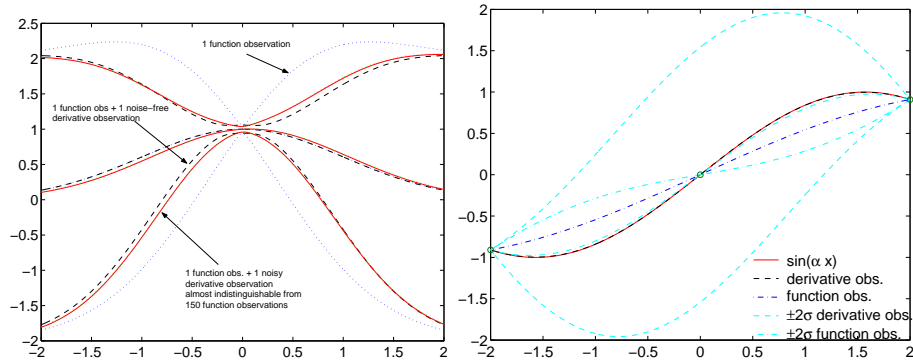

(a) The effect of adding a derivative observation on the prediction uncertainty – standard deviation of GP predictions

(b) Effect of including a noise-free derivative or function observation on the prediction of mean and variance, given appropriate hyperparameters.

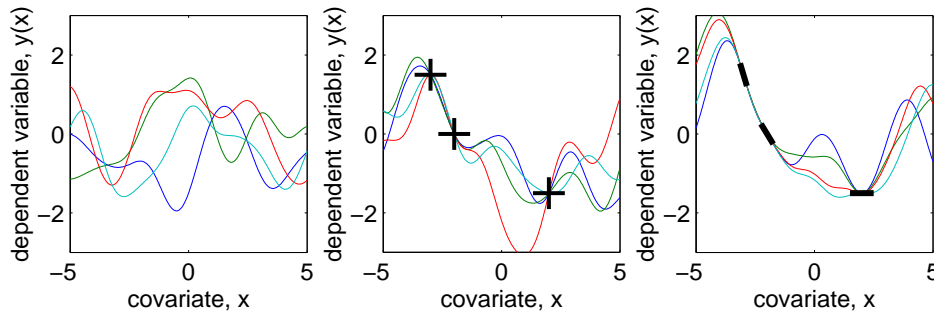

(c) Examples of realisations drawn from a Gaussian process with $\alpha = \gamma_1 = 1$, left – no data, middle, showing the constraining effect of function observations (crosses), and right the effect of function & derivative observations (lines).

Figure 2: Variance effects of derivative information.

Note that the addition of a derivative point does not have an effect on the mean prediction in any of the cases, because the function derivative is zero. The striking effect of the derivative is on the uncertainty. In the case of prediction using function data the uncertainty increases as we move away from the function observation. Addition of a noise-free derivative observation does not affect uncertainty at $x = 0$, but it does mean that uncertainty increases more slowly as we move away from 0, but if uncertainty on the derivative increases, then there is less of an impact on variance. The model based on the single derivative observation identified from the 150 noisy function observations is almost indistinguishable from the model with all 150 function observations.

To further illustrate the effect of adding derivative information, consider the pairs of noise-free observations of $y = \sin(x)$. The hyper-parameters of the model are obtained through a training involving large amounts of data, but we then perform inference using only points at $-2, 0, 2$. For illustration, the function point at $x = 0$ is replaced with a derivative point at the same location, and the results shown in Figure 2(b).

## 3 Nonlinear dynamics example

As an example of a situation where we wish to integrate derivative and function observations we look at a discrete-time nonlinear dynamic system

$$x_{k+1} = x_k - 0.1x_k^3 + 0.1u_k, \tag{10}$$

$$y_{k+1} = x_k + \epsilon_k, \tag{11}$$

where $x_k$ is the system state at time $k$, $y_k$ is the observed output, $u_k$ is the control input and noise term $\epsilon_k \sim \mathcal{N}(0, \sigma^2)$. A standard starting point for identification is to find linear dynamic models at various points on the manifold of equilibria. In the first part of the experiment, we wish to acquire training data by stimulating the system input $u$ to take the system through a wide range of conditions along the manifold of equilibria, shown in Figure 3(a). The linearisations are each identified from 200 function observations $(\mathbf{x_i}, \mathbf{y_i})$ obtained by starting a simulation at $x_0$ and perturbing the control signal about $u_0$ by $\mathcal{N}(0, 0.004)$.

We infer the system response, and the derivative response at various points along the manifold of equilibria, and plot these in Figure 4. The quadratic derivative $\partial y / \partial x$ from the cubic true function is clearly visible in Figure 4(c), and is smooth, despite the presence of several derivative observations with significant errors, because of the appropriate estimates of derivative uncertainty. The $\partial y / \partial u$ is close to constant 0.1 in Figure 4(c). Note that the function 'observations' derived from the linearisations have much lower uncertainty than the individual function observations.

As a second part of the experiment as shown in Figure 3(b), we now add some off-equilibrium function observations to the training set, by applying large control perturbations to the system, taking it through transient regions. We perform a new hyper-parameter optimisation using the using the combination of the transient, off-equilibrium observations and the derivative observations already available. The model incorporates both groups of data and has reduced variance in the off-equilibrium areas. A comparison of simulation runs from the two models with the true data is shown in Figure 5(a), shows the improvement in performance brought by the combination of equilibrium derivatives and off-equilibrium observations over equilibrium information alone. The combined model is almost identical in response to the true system response.

## 4 Conclusions

Engineers are used to interpreting linearisations, and find them a natural way of expressing prior knowledge, or constraints that a data-driven model should conform to. Derivative observations in the form of system linearisations are frequently used in control engineering, and many nonlinear identification campaigns will have linearisations of different operating regions as prior information. Acquiring perturbation data close to equilibrium is relatively easy, and the large amounts of data mean that equilibrium linearisations can be made very accurate. While in many cases we will be able to have accurate derivative observations, they will rarely be noise-free, and the fact that we can so easily include knowledge of derivative or function observation uncertainty is a major benefit of the Gaussian process prior approach. In this paper we used numerical estimates of the full covariance matrix for each linearisation, which were different for every linearisation. The analytic inference of derivative information from a model, and importantly, its uncertainty is potentially of great importance to control engineers designing or validating robust control laws, e.g. [8]. Other applications of models which base decisions on model derivatives will have similar potential benefits.

Local linearisation models around equilibrium conditions are, however, not sufficient for specifying global dynamics. We need observations away from equilibrium in transient regions, which tend to be much sparser as they are more difficult to obtain experimentally,

and the system behaviour tends to be more complex away from equilibrium. Gaussian processes, with robust inference, and input-dependent uncertainty predictions, are especially interesting in sparsely populated off-equilibrium regions. Summarising the large quantities of near-equilibrium data by derivative 'observations' should signficantly reduce the computational problems associated with Gaussian processes in modelling dynamic systems.

We have demonstrated with a simulation of an example nonlinear system that Gaussian process priors can combine derivative and function observations in a principled manner which is highly applicable in nonlinear dynamic systems modelling tasks. Any smoothing procedure involving linearisations needs to satisfy an integrability constraint, which has not been solved in a satisfactory fashion in other widely-used approaches (e.g. multiple model [10], or Takagi-Sugeno fuzzy methods [11]), but which is inherently solved within the Gaussian process formulation. The method scales to higher input dimensions $D$ well, adding only an extra $D$ derivative observations + one function observation for each linearisation. In fact the real benefits may become more obvious in higher dimensions, with increased quantities of training data which can be efficiently summarised by linearisations, and more severe problems in blending local linearisations together consistently.

# References

[1] A. O'Hagan. On curve fitting and optimal design for regression (with discussion). *Journal of the Royal Statistical Society B*, 40:1–42, 1978.

[2] C. K. I. Williams and C. E. Rasmussen. Gaussian processes for regression. In *Neural Information Processing Systems - 8*, pages 514–520, Cambridge, MA, 1996. MIT press.

[3] C. K. I. Williams. Prediction with Gaussian processes: From linear regression to linear prediction and beyond. In M. I. Jordan, editor, *Learning and Inference in Graphical Models*, pages 599–621. Kluwer, 1998.

[4] D. J. C. MacKay. Introduction to Gaussian Processes. NIPS'97 Tutorial notes., 1999.

[5] A. O'Hagan. Some Bayesian numerical analysis. In J. M. Bernardo, J. O. Berger, A. P. Dawid, and A. F. M. Smith, editors, *Bayesian Statistics 4*, pages 345–363. Oxford University Press, 1992.

[6] C. E. Rasmussen. Gaussian processes to speed up Hybrid Monte Carlo for expensive Bayesian integrals. Draft: available at http://www.gatsby.ucl.ac.uk/ edward/pub/gphmc.ps.gz, 2003.

[7] R. Murray-Smith, T. A. Johansen, and R. Shorten. On transient dynamics, off-equilibrium behaviour and identification in blended multiple model structures. In *European Control Conference, Karlsruhe, 1999*, pages BA–14, 1999.

[8] R. Murray-Smith and D. Sbarbaro. Nonlinear adaptive control using non-parametric Gaussian process prior models. In *15th IFAC World Congress on Automatic Control, Barcelona*, 2002.

[9] D. J. Leith, W. E. Leithead, E. Solak, and R. Murray-Smith. Divide & conquer identification: Using Gaussian process priors to combine derivative and non-derivative observations in a consistent manner. In *Conference on Decision and Control*, 2002.

[10] R. Murray-Smith and T. A. Johansen. *Multiple Model Approaches to Modelling and Control*. Taylor and Francis, London, 1997.

[11] T. Takagi and M. Sugeno. Fuzzy identification of systems and its applications for modeling and control. *IEEE Trans. on Systems, Man and Cybernetics*, 15(1):116–132, 1985.

## Acknowledgements

The authors gratefully acknowledge the support of the *Multi-Agent Control* Research Training Network by EC TMR grant HPRN-CT-1999-00107, support from EPSRC grant *Modern statistical approaches to off-equilibrium modelling for nonlinear system control* GR/M76379/01, support from EPSRC grant GR/R15863/01, and Science Foundation Ireland grant 00/PI.1/C067. Thanks to J.Q. Shi and A. Girard for useful comments.

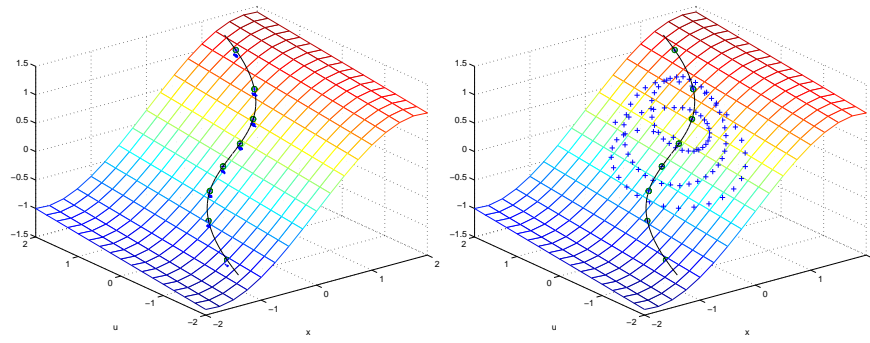

(a) Derivative observations from lin-
earisations identified from the pertur-
bation data. 200 $x, y$ per linearisation
point with noisy $y(\sigma = 0.004)$.

(b) Derivative observations on equilib-
rium, and off-equilibrium function ob-
servations from a transient trajectory.

Figure 3: The manifold of equilibria on the true function. Circles indicate points at which a deriva-
tive observation is made. Crosses indicate a function observation

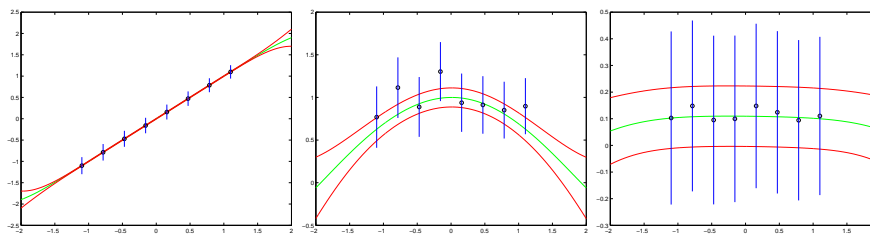

(a) Function observa-
tions

(b) Derivative observa-
tions $\partial y / \partial x$

(c) Derivative observa-
tions $\partial y / \partial u$

Figure 4: Inferred values of function and derivatives, with $\pm 2\sigma$ contours, as $x$ and $u$ are varied
along manifold of equilibria (c.f. Fig. 3) from $x = -2$ to $x = 2$. Circles indicate the locations of the
derivative observations points, lines indicate the uncertainty of observations ($\pm 2$ standard deviations.)

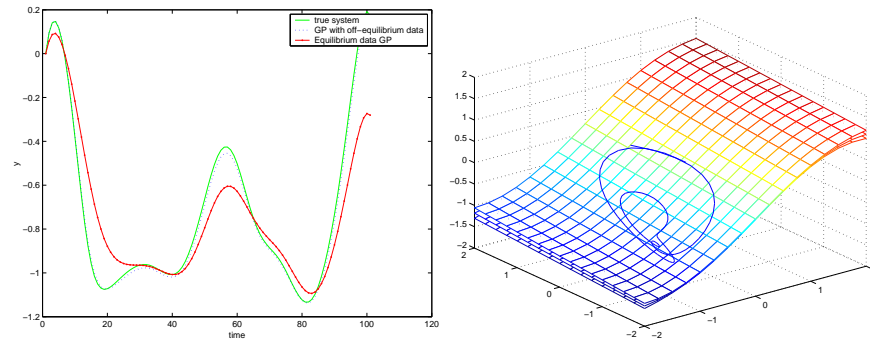

(a) Simulation of dynamics. GP trained
with both on and off-equilibrium data is
close to true system, unlike model based
only on equilibrium data.

(b) Inferred mean and $2\sigma$ surfaces using
linearisations and off-equilibrium data.
The trajectory of the simulation shown
in a) is plotted for comparison.

Figure 5: Modelling results